# Group Sparse Coding with a Laplacian Scale Mixture Prior

**Pierre J. Garrigues**
IQ Engines, Inc.
Berkeley, CA 94704
pierre.garrigues@gmail.com

**Bruno A. Olshausen**
Helen Wills Neuroscience Institute
School of Optometry
University of California, Berkeley
Berkeley, CA 94720
baolshausen@berkeley.edu

## Abstract

We propose a class of sparse coding models that utilizes a Laplacian Scale Mixture (LSM) prior to model dependencies among coefficients. Each coefficient is modeled as a Laplacian distribution with a variable scale parameter, with a Gamma distribution prior over the scale parameter. We show that, due to the conjugacy of the Gamma prior, it is possible to derive efficient inference procedures for both the coefficients and the scale parameter. When the scale parameters of a group of coefficients are combined into a single variable, it is possible to describe the dependencies that occur due to common amplitude fluctuations among coefficients, which have been shown to constitute a large fraction of the redundancy in natural images [1]. We show that, as a consequence of this group sparse coding, the resulting inference of the coefficients follows a divisive normalization rule, and that this may be efficiently implemented in a network architecture similar to that which has been proposed to occur in primary visual cortex. We also demonstrate improvements in image coding and compressive sensing recovery using the LSM model.

## 1 Introduction

The concept of sparsity is widely used in the signal processing, machine learning and statistics communities for model fitting and solving inverse problems. It is also important in neuroscience as it is thought to underlie the neural representations used by the brain. The operation to compute the sparse representation of a signal $x \in \mathbb{R}^n$ with respect to a dictionary of basis functions $\Phi \in \mathbb{R}^{n \times m}$ can be implemented via an $\ell_1$-penalized least-square problem commonly referred to as Basis Pursuit Denoising (BPDN) [2] or Lasso [3]

$$\min_s \frac{1}{2}\|x - \Phi s\|_2^2 + \mu\|s\|_1,  \tag{1}$$

where $\mu$ is a regularization parameter that controls the tradeoff between the quality of the reconstruction and the sparsity. This approach has been applied to problems such as image coding, compressive sensing [4], or classification [5]. The $\ell_1$ penalty leads to solutions where typically a large number of coefficients are exactly zero, which is a desirable property to achieve model selection or data compression, or for obtaining interpretable results. The cost function of BPDN is convex, and many efficient algorithms have been recently developed to solve this problem [6, 7, 8, 9].

Minimizing the cost function of BPDN corresponds to MAP inference in a probabilistic model where the coefficients are independent and have Laplacian priors $p(s_i) = \frac{\lambda}{2}e^{-\lambda|s_i|}$. Hence, the signal model assumed by BPDN is linear, *generative*, and the basis function coefficients are *independent*. In the context of *analysis*-based models of natural images (for a review on *analysis*-based

and *synthesis*-based or *generative* models see [10]), it has been shown that the linear responses of natural images to Gabor-like filters have kurtotic histograms, and that there can be strong dependencies among these responses in the form of common amplitude fluctuations [11, 12, 13, 14]. It has also been observed in the context of *generative* image models that the inferred sparse coefficients exhibit pronounced statistical dependencies [15, 16], and therefore the independence assumption is violated. It has been proposed in block-$\ell_1$ methods to account for dependencies among the coefficients by dividing them into subspaces such that dependencies within the subspaces are allowed, but not across the subspaces [17] . This approach can produce blocking artifacts and has recently been generalized to overlapping subspaces in [18]. Another approach is to only allow certain configurations of active coefficients [19].

We propose in this paper a new class of prior on the basis function coefficients that makes it possible to model their statistical dependencies in a probabilistic *generative* model, whose inferred representations are more sparse than those obtained with the factorial Laplacian prior, and for which we have efficient inference algorithms. Our approach consists of introducing for each coefficient a hyperprior on the inverse scale parameter $\lambda_i$ of the Laplacian distribution. The coefficient prior is thus a mixture of Laplacian distributions which we denote "Laplacian Scale Mixture" (LSM), which is an analogy to the Gaussian scale mixture (GSM) [12]. Higher-order dependencies of feedforward responses of wavelet coefficients [12] or basis functions learned using independent component analysis [14] have been captured using GSMs, and we extend this approach to a *generative* sparse coding model using LSMs.

We define the Laplacian scale mixture in Section 2, and we describe the inference algorithms in the resulting sparse coding models with an LSM prior on the coefficients in Section 3. We present an example of a factorial LSM model in Section 4, and of a non-factorial LSM model in Section 5 that is particularly well suited to signals having the "group sparsity" property. We show that the non-factorial LSM results in a divisive normalization rule for inferring the coefficients. When the groups are organized topographically and the basis is trained on natural images, the resulting model resembles the neighborhood divisive normalization that has been hypothesized to occur in visual cortex. We also demonstrate that the proposed LSM inference algorithm provides superior performance in image coding and compressive sensing recovery.

## 2 The Laplacian Scale Mixture distribution

A random variable $s_i$ is a Laplacian scale mixture if it can be written $s_i = \lambda_i^{-1} u_i$, where $u_i$ has a Laplacian distribution with scale 1, i.e. $p(u_i) = \frac{1}{2}e^{-|u_i|}$, and the multiplier variable $\lambda_i$ is a positive random variable with probability $p(\lambda_i)$. We also suppose that $\lambda_i$ and $u_i$ are independent. Conditioned on the parameter $\lambda_i$, the coefficient $s_i$ has a Laplacian distribution with inverse scale $\lambda_i$, i.e. $p(s_i|\lambda_i) = \frac{\lambda_i}{2}e^{-\lambda_i|s_i|}$. The distribution over $s_i$ is therefore a continuous mixture of Laplacian distributions with different inverse scales, and it can be computed by integrating out $\lambda_i$

$$p(s_i) = \int_0^\infty p(s_i|\ \lambda_i)p(\lambda_i)d\lambda_i = \int_0^\infty \frac{\lambda_i}{2}e^{-\lambda_i|s_i|}p(\lambda_i)d\lambda_i.$$

Note that for most choices of $p(\lambda_i)$ we do not have an analytical expression for $p(s_i)$. We denote such a distribution a Laplacian Scale Mixture (LSM). It is a special case of the Gaussian Scale Mixture (GSM) [12] as the Laplacian distribution can be written as a GSM.

## 3 Inference in a sparse coding model with LSM prior

We propose the linear generative model

$$x = \Phi s + \nu = \sum_{i=1}^m s_i \varphi_i + \nu, \tag{2}$$

where $x \in \mathbb{R}^n$, $\Phi = [\varphi_1, \ldots, \varphi_m] \in \mathbb{R}^{n \times m}$ is an overcomplete transform or basis set, and the columns $\varphi_i$ are its basis functions. $\nu \sim \mathcal{N}(0, \sigma^2 I_n)$ is small Gaussian noise. The coefficients are endowed with LSM distributions. They can be used to reconstruct $x$ and are called the *synthesis* coefficients.

Given a signal $x$, we wish to infer its sparse representation $s$ in the dictionary $\Phi$. We consider in this section the computation of the maximum a posteriori (MAP) estimate of the coefficients $s$ given the input signal $x$. Using Bayes' rule we have $p(s \mid x) \propto p(x \mid s)p(s)$, and therefore the MAP estimate $\hat{s}$ is given by

$$\hat{s} = \arg\min_s \{-\log p(s \mid x)\} = \arg\min_s \{-\log p(x \mid s) - \log p(s)\}. \qquad (3)$$

In general it is difficult to compute the MAP estimate with an LSM prior on $s$ since we do not necessarily have an analytical expression for the log-likelihood $\log p(s)$. However, we can compute the *complete* log-likelihood $\log p(s, \lambda)$ analytically

$$\log p(s, \lambda) = \log p(s \mid \lambda) + \log p(\lambda) = -\lambda_i|s_i| + \log \frac{\lambda_i}{2} + \log p(\lambda).$$

Hence, if we also observed the latent variable $\lambda$, we would have an objective function that can be maximized with respect to $s$. The standard approach in machine learning when confronted with such a problem is the Expectation-Maximization (EM) algorithm, and we derive in this Section an EM algorithm for the MAP estimation of the coefficients. We use Jensen's inequality and obtain the following upper bound on the posterior likelihood

$$-\log p(s \mid x) \leq -\log p(x \mid s) - \int_\lambda q(\lambda) \log \frac{p(s, \lambda)}{q(\lambda)} d\lambda := \mathcal{L}(q, s), \qquad (4)$$

which is true for any probability distribution $q(\lambda)$. Performing coordinate descent in the auxiliary function $\mathcal{L}(q, s)$ leads to the following updates that are usually called the E step and the M step.

$$\textbf{E Step} \qquad q^{(t+1)} = \arg\min_q \mathcal{L}(q, s^{(t)}) \qquad (5)$$

$$\textbf{M Step} \qquad s^{(t+1)} = \arg\min_s \mathcal{L}(q^{(t+1)}, s) \qquad (6)$$

Let $< . >_q$ denote the expectation with respect to $q(\lambda)$. The M Step (6) simplifies to

$$s^{(t+1)} = \arg\min_s \frac{1}{2\sigma^2}\|x - \Phi s\|_2^2 + \sum_{i=1}^m \langle \lambda_i \rangle_{q^{(t+1)}} |s_i|, \qquad (7)$$

which is a least-square problem regularized by a weighted sum of the absolute values of the coefficients. It is a quadratic program very similar to BPDN, and we can therefore use efficient algorithms developed for BPDN that take advantage of the sparsity of the solution. This presents a significant computational advantage over the GSM prior where the inferred coefficients are not exactly sparse.

We have equality in the Jensen inequality if $q(\lambda) = p(\lambda \mid s)$. The inequality (4) is therefore tight for this particular choice of $q$, which implies that the E step reduces to $q^{(t+1)}(\lambda) = p(\lambda \mid s^{(t)})$. Note that in the M step we only need to compute the expectation of $\lambda_i$ with respect to the maximizing distribution in the E step. Hence we only need to compute the sufficient statistics

$$\langle \lambda_i \rangle_{p(\lambda|s^{(t)})} = \int_\lambda \lambda_i\, p(\lambda \mid s^{(t)})d\lambda. \qquad (8)$$

Note that the posterior of the multiplier given the coefficient $p(\lambda \mid s)$ might be hard to compute. We will see in Section 4.1 that it is tractable if the prior on $\lambda$ is factorial and each $\lambda_i$ has a Gamma distribution, as the Laplacian distribution and the Gamma distribution are conjugate. We can apply the efficient algorithms developed for BPDN to solve (7). Furthermore, warm-start capable algorithms are particularly interesting in this context as we can initialize the algorithm with $s^{(t)}$, and we do not expect the solution to change much after a few iterations of EM.

## 4  Sparse coding with a factorial LSM prior

We propose in this Section a sparse coding model where the distribution of the multipliers is factorial, and each multiplier has a Gamma distribution, i.e. $p(\lambda_i) = (\beta^\alpha/\Gamma(\alpha))\lambda_i^{\alpha-1}e^{-\beta\lambda_i}$, where $\alpha$ is the shape parameter and $\beta$ is the inverse scale parameter. With this particular choice of a prior on the multiplier, we can compute the probability distribution of $s_i$ analytically:

$$p(s_i) = \frac{\alpha\beta^\alpha}{2(\beta + |s_i|)^{\alpha+1}}.$$

This distribution has heavier tails than the Laplacian distribution. The graphical model corresponding to this generative model is shown in Figure 1.

### 4.1 Conjugacy

The Gamma distribution and Laplacian distribution are *conjugate*, i.e. the posterior probability of $\lambda_i$ given $s_i$ is also a Gamma distribution when the prior over $\lambda_i$ is a Gamma distribution and the conditional probability of $s_i$ given $\lambda_i$ is a Laplace distribution with inverse scale $\lambda_i$. Hence, the posterior of $\lambda_i$ given $s_i$ is a Gamma distribution with parameters $\alpha + 1$ and $\beta + |s_i|$.

The conjugacy is a key property that we can use in our EM algorithm proposed in Section 3. We saw that the solution of the E step is given by $q^{(t+1)}(\lambda) = p(\lambda \mid s^{(t)})$. In the factorial model we have $p(\lambda \mid s) = \prod_i p(\lambda_i \mid s_i^{(t)})$. The solution of the E step is therefore a product of Gamma distributions with parameters $\alpha + 1$ and $\beta + |s_i^{(t)}|$, and the sufficient statistics (8) are given by

$$\langle \lambda_i \rangle_{p(\lambda_i \mid s_i^{(t)})} = \frac{\alpha + 1}{\beta + |s_i^{(t)}|}. \tag{9}$$

A coefficient that has a small value after $t$ iterations but is not exactly zero will have in the next iteration a large reweighting factor $\lambda_i^{(t+1)}$, which increases the chance that it will be set to zero in the next iteration, resulting in a sparser representation. On the other hand, a coefficient having a large value after $t$ iterations corresponds to a feature that is very salient in the signal $x$. It is therefore beneficial to reduce its corresponding inverse scale $\lambda_i^{(t+1)}$ such that it is not penalized and can account for as much information as possible.

We saw that with the Gamma prior we can compute the distribution of $s_i$ analytically, and therefore we can compute the gradient of $\log p(s \mid x)$ with respect to $s$. Hence another inference algorithm is to descend the cost function in (3) directly using a method such as conjugate gradient, or the method proposed in [20] where the authors also exploit the conjugacy of the Laplacian and Gamma priors. We argue here that the EM algorithm is in fact more efficient. The solution of (7) indeed has typically few elements that are non-zero, and the computational complexity scales with the number of non-zero coefficients [6, 7]. On the other hand, a gradient-based method will have a harder time identifying the support of the solution, and therefore the required computations will involve all the coefficients, which is computationally expensive.

The update formula (9) is coincidentally equivalent to the reweighted L1 minimization scheme proposed by Candès et al. [21]. They solve the following sequence of problems

$$s^{(t+1)} = \arg\min_s \sum_{i=1}^{m} \lambda_i^{(t)} |s_i| \ \text{ subject to } \ \|x - \Phi s\|_2 \leq \delta \tag{10}$$

with update $\lambda_i^{(t+1)} = 1/(\beta + |s_i^{(t)}|)$ (which is identical to our rule when $\alpha = 0$). The authors show that the solutions achieved by their algorithm are more sparse than the solution where $\lambda_i = 1$ for all $i$. Whereas they derive this rule from mathematical intuitions regarding the L1 ball, we show that this update rule follows from from Bayesian inference assuming a Gamma prior over $\lambda$. It was also shown that evidence maximization in a sparse coding model with an automatic relevance determination prior can also be solved via a sequence of reweighted $\ell_1$ optimization problems [22].

### 4.2 Application to image coding

It has been shown that the convex relaxation consisting of replacing the $\ell_0$ norm with the $\ell_1$ norm is able to identify the sparsest solution under some conditions on the dictionary of basis functions [23]. However, these conditions are typically not verified for the dictionaries learned from the statistics of natural images [24]. For instance, it was observed in [16] that it is possible to infer sparser representations with a prior over the coefficients that is a mixture of a delta function at zero and a Gaussian distribution than with the Laplacian prior. We show that our proposed inference algorithm also leads to representations that are more sparse, as the LSM prior with Gamma hyperprior has heavier tails than the Laplacian distribution. We selected 1000 $16 \times 16$ image patches at random, and computed their sparse representations in a dictionary with 256 basis functions using both the conventional Laplacian prior and our LSM prior. The dictionary is learned from the statistics of natural images [24] using a Laplacian prior over the coefficients. To ensure that the reconstruction error is the same in both cases, we solve the constrained version of the problem as in [21], where we require that the signal to noise ratio of the reconstruction is equal to 10. We choose $\beta = 0.01$ and 5

EM iterations. We can see in Figure 2 that the representations using the LSM prior are indeed more sparse by approximately a factor of 2. Note that the computational complexity to compute these sparse representations is much lower than that of [16].

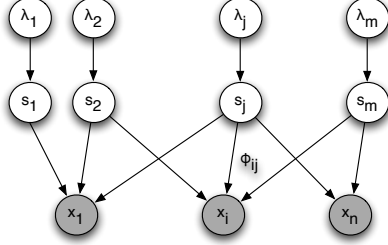

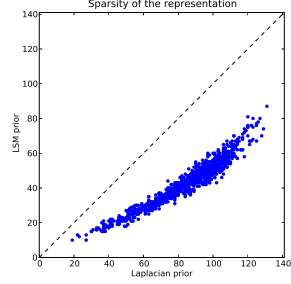

Figure 1: Graphical model representation of our proposed generative model where the multipliers distribution is factorial.

Figure 2: Sparsity comparison. On the x-axis (resp. y-axis) is the $\ell_0$ norm of the representation inferred with the Laplacian prior (resp. LSM prior).

## 5   Sparse coding with a non-factorial model

It has been shown that many natural signals such as sound or images have a particular type of higher-order, sparse structure in which active coefficients occur in groups corresponding to basis functions having similar properties (position, orientation, or frequency tuning) [25, 1]. We focus in this Section on a class of signals that has a particular type of higher-order structure where the active coefficients occur in groups. We show here that the LSM prior can be used to capture this group structure in natural images, and we propose an efficient inference algorithm for this case.

### 5.1   Group sparsity

We consider a dictionary $\Phi$ such that the basis functions can be divided in a set of disjoint groups or neighborhoods indexed by $\mathcal{N}_k$, i.e. $\{1, \ldots, m\} = \bigcup_{k \in \Lambda} \mathcal{N}_k$, and $\mathcal{N}_i \cap \mathcal{N}_j = \emptyset$ if $i \neq j$. A signal having the group sparsity property is such that the sparse coefficients occur in groups, i.e. the indices of the nonzero coefficients are given by $\bigcup_{k \in \Gamma} \mathcal{N}_k$, where $\Gamma$ is a subset of $\Lambda$.

The group sparsity structure can be captured with the LSM prior by having all the coefficients in a group share the same inverse scale parameter, i.e. for all $i \in \mathcal{N}_k$, $\lambda_i = \lambda_{(k)}$. The corresponding graphical model is shown in Figure 3. This addresses the case where dependencies are allowed within groups, but not across groups as in the block-$\ell_1$ method [17]. Note that for some types of dictionaries it is more natural to consider overlapping groups to avoid blocking artifacts. We propose in Section 5.2 inference algorithms for both overlapping and non-overlapping cases.

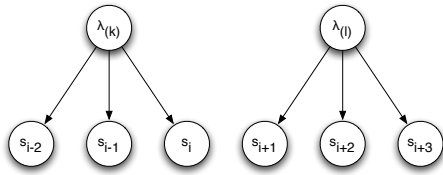

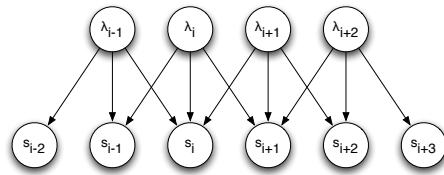

Figure 3: The two groups $\mathcal{N}_{(k)} = \{i-2, i-1, i\}$ and $\mathcal{N}_{(l)} = \{i+1, i+2, i+3\}$ are non-overlapping.

Figure 4: The basis function coefficients in the neighborhood defined by $\mathcal{N}(i) = \{i-1, i, i+1\}$ share the same multiplier $\lambda_i$.

### 5.2   Inference

In the EM algorithm we proposed in Section 3, the sufficient statistics that are computed in the E step are $\langle \lambda_i \rangle_{p(\lambda_i | s^{(t)})}$ for all $i$. We suppose as in Section 4.1 that the prior on $\lambda_{(k)}$ is Gamma with

parameters $\alpha$ and $\beta$. Using the structure of the dependencies in the probabilistic model shown in Figure 3, we have

$$\langle \lambda_i \rangle_{p(\lambda_i | s^{(t)})} = \langle \lambda_{(k)} \rangle_{p(\lambda_{(k)} | s_{\mathcal{N}_k}^{(t)})} \tag{11}$$

where the index $i$ is in the group $\mathcal{N}_k$, and $s_{\mathcal{N}_k} = (s_j)_{j \in \mathcal{N}_k}$ is the vector containing all the coefficients in the group. Using the conjugacy of the Laplacian and Gamma distributions, the distribution of $\lambda_{(k)}$ given all the coefficients in the neighborhood is a Gamma distribution with parameters $\alpha + |\mathcal{N}_k|$ and $\beta + \sum_{j \in N_k} |s_j|$, where $|\mathcal{N}_k|$ denotes the size of the neighborhood. Hence (11) can be rewritten as follows

$$\lambda_{(k)}^{(t+1)} = \frac{\alpha + |\mathcal{N}_k|}{\beta + \sum_{j \in \mathcal{N}_k} |s_j^{(t)}|}. \tag{12}$$

The resulting update rule is a form of divisive normalization. We saw in Section 2 that we can write $s_k = \lambda_{(k)}^{-1} u_k$, where $u_k$ is a Laplacian random variable with scale 1, and thus after convergence we have $u_k^{(\infty)} = (\alpha + |\mathcal{N}_k|) s_k^{(\infty)} / (\beta + \sum_{j \in \mathcal{N}_k} |s_j^{(\infty)}|)$. Such rescaling operations are also thought to play an important role in the visual system. [25]

Now let us consider the case where coefficient neighborhoods are allowed to overlap. Let $\mathcal{N}(i)$ denote the indices of the neighborhood that is centered around $s_i$ (see Figure 4 for an example). We propose to estimate the scale parameter $\lambda_i$ by only considering the coefficients in $\mathcal{N}(i)$, and suppose that they all share the same multiplier $\lambda_i$. In this case the EM update is given by

$$\lambda_i^{(t+1)} = \frac{\alpha + |\mathcal{N}(i)|}{\beta + \sum_{j \in \mathcal{N}(i)} |s_j^{(t)}|}. \tag{13}$$

Note that we have not derived this rule from a proper probabilistic model. A coefficient is indeed a member of many neighborhoods as shown in Figure 4, and the structure of the dependencies implies $p(\lambda_i \mid s) \neq p(\lambda_i \mid s_{N(i)})$. However, we show experimentally that estimating the multiplier using (13) gives good performance. A similar approximation is used in the GSM *analysis*-based model [26]. Note that the noise shaping algorithm, which bears similarities with the iterative thresholding algorithm developed for BPDN [7], is modified in [27] using an update that is essentially inversely proportional to ours. The authors show improved coding efficiency in the context of natural images.

## 5.3 Compressive sensing recovery

In compressed sensing, we observe a number $n$ of random projections of a signal $s_0 \in \mathbb{R}^m$, and it is in principle impossible to recover $s_0$ if $n < m$. However, if $s_0$ has $p$ non-zero coefficients, it has been shown in [28] that it is sufficient to use $n \propto p \log m$ such measurements. We denote by $W \in \mathbb{R}^{n \times m}$ the measurement matrix and let $y = W s_0$ be the observations. A standard method to obtain the reconstruction is to use the solution of the Basis Pursuit (BP) problem

$$\hat{s} = \arg \min_s \|s\|_1 \quad \text{subject to} \quad W s = y. \tag{14}$$

Note that the solution of BP is the solution of BPDN as $\mu$ converges to zero in (1), or $\delta = 0$ in (10). If the signal has structure beyond sparsity, one can in principle recover the signal with even fewer measurements using an algorithm that exploits this structure [19, 29]. We therefore compare the performance of BP with the performance of our proposed LSM inference algorithms

$$s^{(t+1)} = \arg \min_s \sum_{i=1}^m \lambda_i^{(t)} |s_i| \quad \text{subject to} \quad W s = y. \tag{15}$$

We denote by RWBP the algorithm with the factorial update (9), and RW$_3$BP (resp. RW$_5$BP) the algorithm with our proposed divisive normalization update (13) with group size 3 (resp. 5). We consider 50-dimensional signals that are sparse in the canonical basis and where the neighborhood size is 3. To sample such a signal $s \in \mathbb{R}^{50}$, we draw a number $d$ of "centroids" $i$, and we sample three values for $s_{i-1}$, $s_i$ and $s_{i+1}$ using a normal distribution of variance 1. The groups are thus allowed to overlap. A compressive sensing recovery problem is parameterized by $(m, n, d)$. To explore the problem space we display the results using phase plots as in [30], which plots performance as a function of different parameter settings. We fix $m = 50$ and parameterize the phase plots using the indeterminacy of the system indexed by $\delta = n/m$, and the approximate sparsity of the system

indexed by $\rho = 3d/m$. We vary $\delta$ and $\rho$ in the range $[.1, .9]$ using a 30 by 30 grid. For a given value $(\delta, \rho)$ on the grid, we sample 10 sparse signals using the corresponding $(m, n, d)$ parameters. The underlying sparse signal is recovered using the three algorithms and we average the recovery error $\|\hat{s} - s_0\|_2 / \|s_0\|_2$ for each of them. We show in Figure 5 that RW$_3$BP clearly outperforms RWBP. There is a slight improvement by going from BP to RWBP (see supplementary material), but this improvement is rather small as compared with going from RWBP to RW$_3$BP and RW$_5$BP. This illustrates the importance of using the higher-order structure of the signals in the inference algorithm. The performance of RW$_3$BP and RW$_5$BP is comparable (see supplementary material), which shows that our algorithm is not very sensitive to the choice of the neighborhood size.

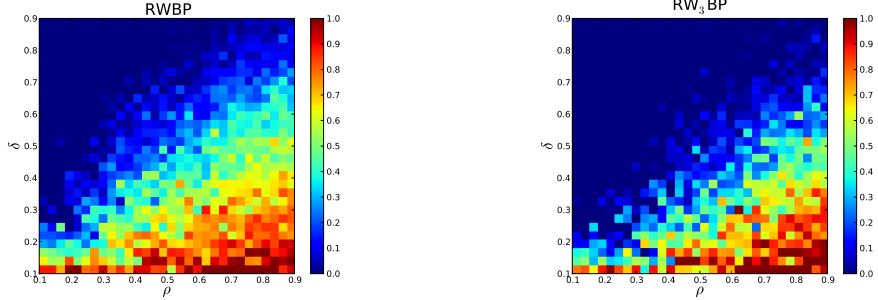

Figure 5: Compressive sensing recovery results using synthetic data. Shown are the phase plots for a sequence of BP problems with the factorial update (RWBP), and a sequence of BP problems with the divisive normalization update with neighborhood size 3 (RW$_3$BP). On the x-axis is the sparsity of the system indexed by $\rho = 3d/m$, and on the y-axis is the indeterminacy of the system indexed by $\delta = n/m$. At each point $(\rho, \delta)$ in the phase plot we display the average recovery error.

## 5.4 Application to natural images

It has been shown that adapting a dictionary of basis functions to the statistics of natural images so as to maximize sparsity in the coefficients results in a set of dictionary elements whose spatial properties match those of V1 (primary visual cortex) receptive fields [24]. However, the basis functions are learned under a probabilistic model where the probability density over the basis functions coefficients is factorial, whereas the sparse coefficients exhibit statistical dependencies [15, 16]. Hence, a generative model with factorial LSM is not rich enough to capture the complex statistics of natural images. We propose here to model these dependencies using a non-factorial LSM model. We fix a topography where the basis functions coefficients are arranged on a 2D grid, and with overlapping neighborhoods of fixed size $3 \times 3$. The corresponding inference algorithm uses the divisive normalization update (13).

We learn the optimal dictionary of basis functions $\Phi$ using the learning rule $\Delta \Phi = \eta \left\langle (x - \Phi \hat{s}) \hat{s}^T \right\rangle$ as in [24], where $\eta$ is the learning rate, $\hat{s}$ are the basis functions coefficients inferred under the model (13), and the average is taken over a batch of size 100. We fix $n = m = 256$, and sample $16 \times 16$ image patches from a set of whitened images, using a total of 100000 batches. The learned basis functions are shown in Figure 6. We see here that the neighborhoods of size $3 \times 3$ group basis functions at a similar position, scale and orientation. The topography is similar to how neurons are arranged in the visual cortex, and is reminiscent of the results obtained in topographic ICA [13] and topographic mixture of experts models [31]. An important difference is that our model is based on a *generative* sparse coding model in which both inference and learning can be implemented via local network interactions [7]. Because of the topographic organization, we also obtain a neighborhood-based divisive normalization rule.

Does the proposed non-factorial model represent image structure more efficiently than those with factorial priors? To answer this question we measured the models' ability to recover sparse structure in the compressed sensing setting. We note that the basis functions are learned such that they represent the sparse structure in images, as opposed to representing the images exactly (there is a noise term in the generative model (2)). Hence, we design our experiment such that we measure the recovery of this sparse structure. Using the basis functions shown in Figure 6, we first infer the

sparse coefficients $s_0$ of an image patch $x$ such that $\|x - \Phi s_0\|_2 < \delta$ using the inference algorithm corresponding to the model. We fix $\delta$ such that the SNR is 10, and thus the three sparse approximations for the three models contain the same amount of signal power. We then compute random projections $y = \tilde{W}\Phi s_0$ where $\tilde{W}$ is the random measurements matrix. We attempt to recover the sparse coefficients as in Section 5.3 by substituting $W := \Phi\tilde{W}$, and $y := \Phi s_0$. We compare the recovery performance $\|\Phi\hat{s} - \Phi s_0\|_2 / \|\Phi s_0\|_0$ for 100 $16 \times 16$ image patches selected at random, and we use 110 random projections. We can see in Figure 7 that the model with non-factorial LSM prior outperforms the other models as it is able to capture the group sparsity structure in natural images.

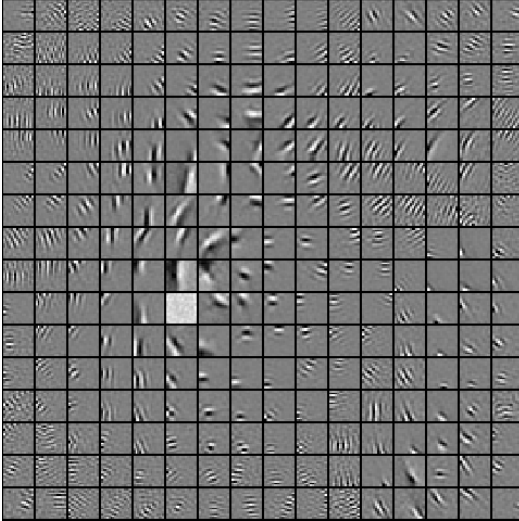

Figure 6: Basis functions learned in a non-factorial LSM model with overlapping groups of size $3 \times 3$

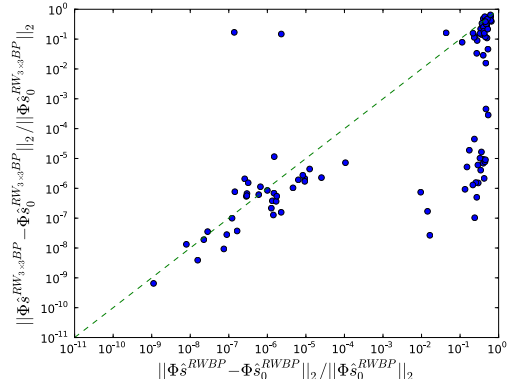

Figure 7: Compressive sensing recovery. On the x-axis is the recovery performance for the factorial LSM model (RWBP), and on the y-axis the recovery performance for the non-factorial LSM model with $3 \times 3$ overlapping groups (RW$_{3\times3}$BP). RW$_{3\times3}$BP outperforms RWBP. See supplementary material for the comparison between RW$_{3\times3}$BP and BP as well as between RWBP and BP.

## 6 Conclusion

We introduced a new class of probability densities that can be used as a prior for the coefficients in a *generative* sparse coding model of images. By exploiting the conjugacy of the Gamma and Laplacian prior, we were able to derive an efficient inference algorithm that consists of solving a sequence of reweighted $\ell_1$ least-square problems, thus leveraging the multitude of algorithms already developed for BPDN. Our framework also makes it possible to capture higher-order dependencies through group sparsity. When applied to natural images, the learned basis functions of the model may be topographically organized according to the specified group structure. We also showed that exploiting the group sparsity results in performance gains for compressive sensing recovery on natural images. An open question is the learning of group structure, which is a topic of ongoing work.

We wish to acknowledge support from NSF grant IIS-0705939.

## References

[1] S. Lyu and E. P. Simoncelli. Statistical modeling of images with fields of gaussian scale mixtures. In *Advances in Neural Computation Systems (NIPS)*, Vancouver, Canada, 2006.

[2] S.S. Chen, D.L. Donoho, and M.A. Saunders. Atomic decomposition by basis pursuit. *SIAM Journal on Scientific Computing*, 20(1):33–61, 1999.

[3] R. Tibshirani. Regression shrinkage and selection via the lasso. *Journal of the Royal Statistical Society. Series B*, 58(1):267–288, 1996.

[4] Y. Tsaig and D.L. Donoho. Extensions of compressed sensing. *Signal Processing*, 86(3):549–571, 2006.

[5] R. Raina, A. Battle, H. Lee, B. Packer, and A.Y. Ng. Self-taught learning: Transfer learning from unlabeled data. *Proceedings of the Twenty-fourth International Conference on Machine Learning*, 2007.

[6] B. Efron, T. Hastie, I. Johnstone, and R. Tibshirani. Least angle regression. *Annals of Statistics*, 32(2):407–499, 2004.

[7] C.J. Rozell, D.H Johnson, R.G. Baraniuk, and B.A. Olshausen. Sparse coding via thresholding and local competition in neural circuits. *Neural Computation*, 20(10):2526–2563, October 2008.

[8] J. Friedman, T. Hastie, H. Hoefling, and R. Tibshirani. Pathwise coordinate optimization. *The Annals of Applied Statistics*, 1(2):302–332, 2007.

[9] M. Figueiredo, R. Nowak, and S. Wright. Gradient projection for sparse reconstruction: Application to compressed sensing and other inverse problems. *IEEE Journal of Selected Topics in Signal Processing*, 1(4):586–597, 2007.

[10] M. Elad, P. Milanfar, and R. Rubinstein. Analysis vs synthesis in signal priors. *Inverse Problems*, 23(3):947–968, June 2007.

[11] C. Zetzsche, G. Krieger, and B. Wegmann. The atoms of vision: Cartesian or polar? *Journal of the Optical Society of America A*, 16(7):1554–1565, 1999.

[12] M.J. Wainwright, E.P. Simoncelli, and A.S. Willsky. Random cascades on wavelet trees and their use in modeling and analyzing natural imagery. *Applied and Computational Harmonic Analysis*, 11(1), July 2001.

[13] A. Hyvärinen, P.O. Hoyer, and M. Inki. Topographic independent component analysis. *Neural Computation*, 13(7):1527–1558, 2001.

[14] Y. Karklin and M.S. Lewicki. A hierarchical bayesian model for learning nonlinear statistical regularities in nonstationary natural signals. *Neural Computation*, 17(2):397–423, February 2005.

[15] P. Hoyer and A. Hyvärinen. A multi-layer sparse coding network learns contour coding from natural images. *Vision Research*, 42:1593–1605, 2002.

[16] P.J. Garrigues and B.A. Olshausen. Learning horizontal connections in a sparse coding model of natural images. In *Advances in Neural Computation Systems (NIPS)*, Vancouver, Canada, 2007.

[17] M. Yuan and Y. Lin. Model selection and estimation in regression with grouped variables. *Journal of the Royal Statistical Society: Series B (Statistical Methodology)*, 68(1):49–67, February 2006.

[18] L. Jacob, G. Obozinski, and J.-P. Vert. Group lasso with overlap and graph lasso. In *International Conference on Machine Learning (ICML)*, 2009.

[19] R.G. Baraniuk, V. Cevher, M.F. Duarte, and C. Hegde. Model-based compressive sensing. *Preprint*, August 2008.

[20] I. Ramirez, F. Lecumberry, and G. Sapiro. Universal priors for sparse modeling. *CAMPSAP*, December 2009.

[21] E.J. Candès, M.B. Wakin, and S.P. Boyd. Enhancing sparsity by reweighted l1 minimization. *J. Fourier Anal. Appl.*, to appear, 2008.

[22] D. Wipf and S. Nagarajan. A new view of automatic relevance determination. In *Advances in Neural Information Processing Systems 20*, 2008.

[23] J.A. Tropp. Just relax: convex programming methods for identifying sparse signals in noise. *IEEE Transactions on Information Theory*, 52(3):1030–1051, 2006.

[24] B.A. Olshausen and D.J. Field. Emergence of simple-cell receptive field properties by learning a sparse code for natural images. *Nature*, 381(6583):607–609, June 1996.

[25] M.J. Wainwright, O. Schwartz, and E.P. Simoncelli. Natural image statistics and divisive normalization: Modeling nonlinearity and adaptation in cortical neurons. In R. Rao, B.A. Olshausen, and M.S. Lewicki, editors, *Statistical Theories of the Brain*. MIT Press, 2001.

[26] J. Portilla, V. Strela, M.J Wainwright, and E.P. Simoncelli. Image denoising using scale mixtures of gaussians in the wavelet domain. *IEEE Transactions on Image Processing*, 12(11):1338–1351, 2003.

[27] R.M. Figueras and E.P. Simoncelli. Statistically driven sparse image representation. In *Proc 14th IEEE Int'l Conf on Image Proc*, volume 6, pages 29–32, September 2007.

[28] E. Candès. Compressive sampling. *Proceedings of the International Congress of Mathematicians*, 2006.

[29] V. Cevher, , M. F. Duarte, C. Hegde, and R. G. Baraniuk. Sparse signal recovery using markov random fields. In *Advances in Neural Computation Systems (NIPS)*, Vancouver, B.C., Canada, 2008.

[30] D. Donoho and Y. Tsaig. Fast solution of l 1-norm minimization problems when the solution may be sparse. *preprint*, 2006.

[31] S. Osindero, M. Welling, and G.E. Hinton. Topographic product models applied to natural scene statistics. *Neural Computation*, 18(2):381–414, 2006.

